# One Sketch For All: Theory and Application of Conditional Random Sampling

**Ping Li**
Dept. of Statistical Science
Cornell University
pingli@cornell.edu

**Kenneth W. Church**
Microsoft Research
Microsoft Corporation
church@microsoft.com

**Trevor J. Hastie**
Dept. of Statistics
Stanford University
hastie@stanford.edu

## Abstract

Conditional Random Sampling (CRS) was originally proposed for efficiently computing pairwise ($l_2$, $l_1$) distances, in *static*, *large-scale*, and *sparse* data. This study modifies the original CRS and extends CRS to handle *dynamic* or *streaming* data, which much better reflect the real-world situation than assuming static data. Compared with many other *sketching* algorithms for dimension reductions such as *stable random projections*, CRS exhibits a significant advantage in that it is "one-sketch-for-all." In particular, we demonstrate the effectiveness of CRS in efficiently computing the Hamming norm, the Hamming distance, the $l_p$ distance, and the $\chi^2$ distance. A generic estimator and an approximate variance formula are also provided, for approximating any type of distances.

We recommend CRS as a promising tool for building highly scalable systems, in machine learning, data mining, recommender systems, and information retrieval.

## 1 Introduction

Learning algorithms often assume a data matrix $\mathbf{A} \in \mathbb{R}^{n \times D}$ with $n$ observations and $D$ attributes and operate on the data matrix $\mathbf{A}$ through pairwise distances. The task of computing and maintaining distances becomes non-trivial, when the data (both $n$ and $D$) are large and possibly dynamic.

For example, if $\mathbf{A}$ denotes a term-doc matrix at Web scale with each row representing one Web page, then $n \approx O(10^{10})$ (which may be verified by querying "A" or "The" in a search engine). Assuming $10^5$ English words, the simplest uni-gram model requires the dimension $D \approx O(10^5)$; and a bi-gram model can boost the dimension to $D \approx O(10^{10})$. Google book search program currently provides data sets on indexed digital books up to **five-grams**. Note that the term-doc matrix is "transposable," meaning that one can treat either documents or terms as features, depending on applications.

Another example is the image data. The *Caltech 256* benchmark contains $n = 30,608$ images, provided by two commercial firms. Using pixels as features, a $1024 \times 1024$ color image can be represented by a vector of dimension $D = 1024^2 \times 3 = 3,145,728$. Using histogram-based features (e.g., [3]), $D = 256^3 = 16,777,216$ is possible if one discretizes the RGB space into $256^3$ scales.

Text data are **large** and **sparse**, as most terms appear only in a small fraction of documents. For example, a search engine reports $10^7$ pagehits for the query "NIPS," which is not common to the general audience. Out of $10^{10}$ pages, $10^7$ pagehits indicate a *sparsity* of 99.9%. (We define *sparsity* as the percentage of zero elements.) In the absolute magnitude, however, $10^7$ is actually very large.

Not all large-scale data are sparse. Image data are usually sparse when features are represented by histograms; they are, however, dense when pixel-based features are used.

### 1.1 Pairwise Distances Used in Machine Learning

The $l_p$ distance and $\chi^2$ distance are both popular. Denote by $u_1$ and $u_2$ the leading two rows in $\mathbf{A} \in \mathbb{R}^{n \times D}$. The $l_p$ distance (raised to the $p$th power), and the $\chi^2$ distance, are, respectively,

$$d_p(u_1, u_2) = \sum_{i=1}^{D} |u_{1,i} - u_{2,i}|^p, \qquad d_{\chi^2}(u_1, u_2) = \sum_{i=1}^{D} \frac{(u_{1,i} - u_{2,i})^2}{u_{1,i} + u_{2,i}}, \qquad \left(\frac{0}{0} = 0\right).$$

The $\chi^2$ distance is only a special case of Helbertian metrics, defined as,

$$d_{H,\alpha,\beta}(u_1, u_2) = \sum_{i=1}^{D} \frac{2^{1/\beta}\left(u_{1,i}^\alpha + u_{2,i}^\alpha\right)^{1/\alpha} - 2^{1/\alpha}\left(u_{1,i}^\beta + u_{2,i}^\beta\right)^{1/\beta}}{2^{1/\alpha} - 2^{1/\beta}}, \quad \alpha \in [1, \infty),\ \beta \in [1/2, \alpha]\ \text{or}\ \beta \in [-\infty, -1].$$

Helbertian metrics are defined over probability space[7] and hence suitable for data generated from histograms, e.g., the "bag-of-words" model. For applications in text and images using SVM, empirical studies have demonstrated the superiority of Helbertian metrics over $l_p$ distances[3, 7, 9].

More generally, we are interested in any linear summary statistics which can be written in the form:

$$d_g(u_1, u_2) = \sum_{i=1}^{D} g(u_{1,i}, u_{2,i}), \tag{1}$$

for any generic function $g$. An efficient method for computing (1) for any $g$ would be desirable.

## 1.2 Bottleneck in Distance/Kernel-based Learning Algorithms

A ubiquitous task in learning is to compute, store, update, and retrieve various types of *distances*[17]. For popular kernel SVM solvers including the SMO algorithm[16], storing and computing kernels is the major bottleneck[2], because computing kernels is expensive, and more seriously, storing the full kernel matrix in memory is infeasible when the number of observations $n > 10^5$.

One popular strategy is to evaluate kernels **on the fly**[2]. This works well in low-dimensional data (i.e., relatively small $D$). With high-dimensional data, however, either computing distances on-demand becomes too slow or the data matrix $\mathbf{A} \in \mathbb{R}^{n \times D}$ itself may not fit in memory.

We should emphasize that this challenge is a universal issue in distance-based methods, not limited to SVMs. For example, popular clustering algorithms and multi-dimensional scaling algorithms require frequently accessing a (di)similarity matrix, which is usually distance-based.

In addition to computing and storing distances, another general issue is that, for many real-world applications, entries of the data matrix may be frequently updated, for example, data streams[15]. There have been considerable studies on learning from dynamic data, e.g., [5, 1]. Since streaming data are often not stored (even on disks), computing and updating distances becomes challenging.

## 1.3 Contributions and Paper Organization

*Conditional Random Sampling (CRS)*[12, 13] was originally proposed for efficiently computing pairwise ($l_2$ and $l_1$) distances, in large-scale static data. The contributions of this paper are:

1. We extend CRS to handle dynamic data. For example, entries of a matrix may vary over time, or the data matrix may not be stored at all. We illustrate that CRS has the **one-sketch-for-all** property, meaning that the same set of samples/sketches can be used for computing any linear summary statistics (1). This is a significant advantage over many other dimension reduction or data stream algorithms. For example, the method of *stable random projections (SRP)*[8, 10, 14] was designed for estimating the $l_p$ norms/distances for a fixed $p$ with $0 < p \le 2$. Recently, a new method named *Compressed Counting*[11] is able to very efficiently approximate the $l_p$ moments of data streams when $p \approx 1$.

2. We introduce a modification to the original CRS and theoretically justify that this modification makes CRS rigorous, at least for computing the Hamming norm, an important application in databases. We point out the original CRS was based on a heuristic argument.

3. We apply CRS for computing Hilbertian metrics[7], a popular family of distances for constructing kernels in SVM. We focus on a special case, by demonstrating that CRS is effective in approximating the $\chi^2$ distance.

Section 2 reviews the original CRS. Section 3 extends CRS to dynamic/streaming data. Section 4 focuses on using CRS to estimate the Hamming norm of a single vector, based on which Section 5 provides a generic estimation procedure for CRS, for estimating any linear summary statistics, with the focus on the Hamming distance and the $\chi^2$ distance. Finally, Section 6 concludes the paper.

## 2 Conditional Random Sampling (CRS), the Original Version

*Conditional Random Sampling (CRS)*[12, 13] is a **local** sampling strategy. Since distances are *local* (i.e., one pair at a time), there is no need to consider the whole matrix at one time.

As the first step, CRS applies a random permutation on the columns of $\mathbf{A} \in \mathbb{R}^{n \times D}$. Figure 1(a) provides an example of a column-permuted data matrix. The next step of CRS is to construct a

*sketch* for each row of the data matrix. A sketch can be viewed as a linked list which stores a small fraction of the non-zero entries from the front of each row. Figure 1(b) demonstrates three sketches corresponding to the three rows of the (column) permuted data matrix in Figure 1(a).

| | 1 | 2 | 3 | 4 | 5 | 6 | 7 | 8 | 9 | 10 | 11 | 12 | 13 | 14 | 15 | 16 |
|-----|---|---|---|---|---|---|---|---|---|----|----|----|----|----|----|----|
| $u_1$ | 5 | 0 | 0 | 1 | 0 | 7 | 0 | 0 | 0 | 8 | 0 | 1 | 0 | 8 | 0 | 2 |
| $u_2$ | 0 | 9 | 2 | 0 | 6 | 0 | 0 | 7 | 0 | 5 | 0 | 0 | 4 | 0 | 0 | 13 |
| $u_3$ | 0 | 4 | 0 | 0 | 2 | 0 | 0 | 0 | 8 | 0 | 0 | 3 | 0 | 0 | 12 | 0 |

$K_1$: 1 {5}  4 {1}  6 {7}  10 {8}
$K_2$: 2 {9}  3 {2}  5 {6}  8 {7}
$K_3$: 2 {4}  5 {2}  9 {8}  12 {3}

(a) Permuted data matrix        (b) Sketches

Figure 1: (a): A data matrix with three rows and $D = 16$ columns. We assume the columns are already permuted. (b): Sketches are the first $k_i$ non-zero entries ascending by IDs (here $k_i = 4$).

In Figure 1, the sketch for row $u_i$ is denoted by $K_i$. Each element of $K_i$ is a tuple "ID {val}," where "ID" is the column ID after the permutation and "{val}" is the value of that entry.

Consider two rows $u_1$ and $u_2$. The last (largest) IDs of sketches $K_1$ and $K_2$ are $\max(\text{ID}(K_1)) = 10$ and $\max(\text{ID}(K_2)) = 8$, respectively. Here, "ID(K)" stands for the vector of IDs in the sketch K. It is clear that $K_1$ and $K_2$ contain all information about $u_1$ and $u_2$ from columns 1 to $\min(10, 8) = 8$. Had we directly taken the first $D_s = 8$ columns from the permuted data matrix, we would obtain the same non-zero entries as in $K_1$ and $K_2$, if we exclude elements in $K_1$ and $K_2$ whose IDs $> D_s = 8$. in this example, the element 10{8} in sketch $K_1$ is excluded.

On the other hand, since the columns are already permuted, any $D_s$ columns constitute a random sample of size $D_s$. This means, by only looking at sketches $K_1$ and $K_2$, one can obtain a "random" sample of size $D_s$. By statistics theory, one can easily obtain an **unbiased** estimate of any linear summary statistics from a random sample. Since $D_s$ is unknown until we look at $K_1$ and $K_2$ together, [13] viewed this as a random sample **conditioning** on $D_s$.

Note that the $D_s$ varies pairwise. When considering the rows $u_1$ and $u_3$, the sketches $K_1$ and $K_3$ suggest their $D_s$ = $\min(\max(\text{ID}(K_1)), \max(\text{ID}(K_3)))$ = $\min(10,12) = 10$.

In this study, we point out that, although the "conditioning" argument appeared intuitive, it is only a (good) heuristic. There are two ways to understand why this argument is not strictly correct.

Consider a *true* random sample of size $D_s$, directly obtained from the first $D_s$ columns of the permuted data matrix. Assuming sparse data, elements at the $D_s$th column should be most likely zero. However, in the "conditional random sample" obtained from CRS, at least one element at the $D_s$th column is non-zero. Thus, the estimates of the original CRS are, strictly speaking, **biased**.

For a more obvious example, we can consider two rows with exactly one non-zero entry in each row at the same column. The original CRS can not obtain an unbiased estimate unless $D_s = D$.

## 3   CRS for Dynamic Data and Introduction to Stable Random Projections

The original CRS was proposed for static data. In reality, the "data matrix" may be frequently updated. When data arrive in a streaming fashion, they often will not be stored (even on disks)[15]. Thus, a one-pass algorithm is needed to compute and update distances for training. Learning with dynamic (or incremental) data has become an active topic of research, e.g., [5, 1].

### 3.1   Dynamic/Streaming Data

We first consider only one data vector $u$ of length $D$ (viewed as one row in the data matrix). At each time $t$, there is an input stream $s_t = (i_t, I_t)$, $i_t \in [1, \ D]$ which updates $u$ (denoted by $u_t$) by

$$u_t[i_t] = \text{H}(u_{t-1}[i_t], I_t),$$

where $I_t$ is the increment/decrement at time $t$ and H is an updating function. The so-called *Turnstile* model [15] is extremely popular and assumes a linear updating function H, i.e.,

$$u_t[i_t] = u_{t-1}[i_t] + I_t. \tag{2}$$

For example, $u_t[i_t]$ can represent the number of orders a "user" $i$ has purchased up to time $t$, where a user may be identified by his/her IP address (i.e., $i \in [1, D = 2^{64}]$); $I_t$ is the number of orders the user $i$ orders (i.e., $I_t > 0$) or cancels (i.e., $I_t < 0$) at time $t$.

In terms of the data matrix $\mathbf{A} \in \mathbb{R}^{n \times D}$, we can view it to be a collection of $n$ data streams.

## 3.2 CRS for Streaming Data

For each stream $u_t$, we maintain a sketch K with length (i.e., capacity) $k$. Each entry of K is a tuple "ID{val}." Initially, all entries are empty. The procedure for sketch construction works as follows:

1. Generate a random permutation $\pi : [1, D] \rightarrow [1, D]$.
2. For each $s_t = (i_t, I_t)$, if $\pi[i_t] > \max(\text{ID}(\text{K}))$ and the capacity of K is reached, do nothing.
3. Suppose $\pi[i_t] \leq \max(\text{ID}(\text{K}))$ or the capacity of K is not reached. If an entry with ID = $\pi[i_t]$ does not exist, insert a new entry. Otherwise, update that entry according to $\mathbf{H}$.[1]
4. Apply the procedure to each data stream using the same random permutation mapping $\pi$.

Once sketches are constructed, the estimation procedure will be the same regardless whether the original data are dynamic or static. Thus, we will use static data to verify some estimators of CRS.

## 3.3 (Symmetric) Stable Random Projections (SRP)

Since the method of *(symmetric) stable random projections (SRP)*[8, 10] has become a standard algorithm for data stream computations, we very briefly introduce SRP for the sake of comparisons.

The procedure of SRP is to multiply the data matrix $\mathbf{A} \in \mathbb{R}^{n \times D}$ by a random matrix $\mathbf{R} \in \mathbb{R}^{D \times k}$, whose entries are i.i.d. samples from a standard (symmetric) stable distribution $S(p, 1), 0 < p \leq 2$.

Consider two rows, $u_1$ and $u_2$, in $\mathbf{A}$. By properties of stable distributions, the projected vectors $v_1 = \mathbf{R}^{\mathrm{T}} u_1$ and $v_2 = \mathbf{R}^{\mathrm{T}} u_2$ have i.i.d. stable entries, i.e., for $j = 1$ to $k$,

$$v_{1,j} \sim S\left(p, F_p = \sum_{i=1}^{D} |u_{1,i}|^p\right), \qquad v_{1,j} - v_{2,j} \sim S\left(p, d_p = \sum_{i=1}^{D} |u_{1,i} - u_{2,i}|^p\right).$$

Thus, one can estimate an individual norm or distance from $k$ samples. SRP is applicable to dynamic/streaming data, provided the data follow the *Turnstile* model in (2). Because the *Turnstile* model is linear and matrix multiplication is also linear, one can conduct $\mathbf{A} \times \mathbf{R}$ incrementally.

Compared with *Conditional Random Sampling (CRS)*, SRP has an elegant mathematical derivation, with various interesting estimators and rigorous sample complexity bounds, i.e., $k$ can be predetermined in fully rigorous fashion. The accuracy of SRP is not affected by heavy-tailed data.

CRS, however, exhibits certain advantages over SRP:

- *CRS is "one-sketch-for-all".* The same sketch of CRS can approximate any linear summary statistics (1). SRP is limited to the $l_p$ norm and distance with $0 < p \leq 2$. One has to conduct SRP 10 times (and store 10 sets of sketches) if 10 different $p$ values are needed.
- *CRS allows "term-weighting" in dynamic data.* In machine learning, the distances are often computed using weighted data (e.g., $\sqrt{u_{1,i}}$ or $\log(1 + u_{1,i})$), which is critical for good performance. For static data, one can first term-weight the data before applying SRP. For dynamic data, however, there is no way to trace back the original data after projections.
- *CRS is not restricted to the Turnstile model.*
- *CRS is not necessary less accurate*, especially for sparse data or binary data.

## 4 Approximating Hamming Norms in Dynamic Data

Counting the Hamming norm (i.e., number of non-zeros) in an exceptionally long, dynamic vector has important applications[4, 15]. For example, if a vector $u_t$ records the numbers of items users have ordered, one meaningful question to ask may be " how many distinct users are there?"

The purpose of this section is three-fold. (1) This is the case we can rigorously analyze CRS and propose a truly unbiased estimator. (2) This analysis brings better insights and more reasonable estimators for pairs of data vectors. (3) In this case, despite its simplicity, CRS theoretically achieves similar accuracy as stable random projections (SRP). Empirically, CRS (slightly) outperforms SRP.

## 4.1 The Proposed (Unbiased) Estimator and Variance

Suppose we have obtained the sketch K. For example, consider the first row in Figure 1: $D = 16$, $k = 4$ and the number of non-zeros $f = 7$. Lemma 1 (whose proof is omitted) proposes an unbiased estimator of $f$, denoted by $\hat{f}$, and a biased estimator based on the maximum likelihood, $f_{mle}$.

**Lemma 1**

$$\hat{f} = \frac{D(k-1)}{Z-1}, \qquad Z = \max(ID(K)), \qquad E\left(\hat{f}\right) = f, \qquad D \geq f \geq k > 1$$

$$Var\left(\hat{f}\right) < V_f^U = \frac{f^2 - f}{k-2}\frac{D}{D-1} - \frac{(D-f)f}{D-1}, \qquad (k > 2)$$

$$Var\left(\hat{f}\right) > V_f^L = V_f^U - \frac{(k-1)f(f-1)(f-2)D}{(k-2)(k-3)(D-1)(D-2)}, \qquad (k > 3).$$

*Assume $f/D$ is small and $k/f$ is also small, then $Var\left(\hat{f}\right) = \frac{f^2}{k} + O\left(\frac{1}{k^2}\right)$.*

*The maximum likelihood estimator is $\hat{f}_{mle} = \frac{k(D+1)}{Z} - 1$.*

Note that, since $Var\left(\hat{f}\right)/f^2 \approx 1/k$, independent of the data, the estimator $\hat{f}$ actually has the worst-case complexity bound similar to that of SRP[10], although the precise constant is not easy to obtain.

## 4.2 The Approximation Using the Conditioning Argument

Interestingly, this estimator, $\hat{f} = \frac{D(k-1)}{\max(ID(K))-1}$, appears to be the estimator for a hypergeometric random sample of size $D_s = \max(ID(K)) - 1$. That is, suppose we randomly pick $D_s$ balls (without replacement) from a pool of $D$ balls and we observe that $k'$ balls are red; then a natural (and unbiased) estimator for the total number of red balls would be $\frac{D}{D_s}k'$; here $k' = k - 1$.

This seems to imply that the "conditioning" argument in the original CRS in Section 2 is "correct" if we make a simple modification by using the $D_s$ which is the original $D_s$ minus 1. While this is what we will recommend as the modified CRS, it is only a close approximation.

Consider $\hat{f}_{app} = \hat{f}$, where we assume $\hat{f}_{app}$ is the estimator for the hypergeometric distribution, then

$$Var\left(\hat{f}_{app}|D_s = Z - 1\right) = \frac{D^2}{D_s^2}D_s\frac{f}{D}\left(1 - \frac{f}{D}\right) \times \frac{D - D_s}{D-1} = \frac{D}{D-1}\left(\frac{D}{D_s} - 1\right)f\left(1 - \frac{f}{D}\right)$$

$$Var\left(\hat{f}_{app}\right) = E\left(Var\left(\hat{f}_{app}|D_s\right)\right) = \frac{D}{D-1}\left(E\left(\frac{D}{Z-1}\right) - 1\right)f\left(1 - \frac{f}{D}\right) = \frac{Df}{D-1}\left(\frac{f}{k-1} - 1\right)\left(1 - \frac{f}{D}\right) \quad (3)$$

## 4.3 Comparisons with Stable Random Projections (SRP)

Based on the observation that $f = \lim_{p \to 0+}\sum_{i=1}^D |u_i|^p$, [4] proposed using SRP to approximate the $l_p$ norm with very small $p$, as an approximation to $f$. For $p \to 0+$, the recent work for SRP [10] proposed the *harmonic mean* estimator. Recall that after projections $v = \mathbf{R}^T u \in \mathbb{R}^k$ consists of i.i.d. stable samples with scale parameter $F_p = \sum_{i=1}^D |u_i|^p$. The *harmonic mean* estimator is

$$\hat{F}_{p,hm} = \frac{-\frac{2}{\pi}\Gamma(-p)\sin\left(\frac{\pi}{2}p\right)}{\sum_{j=1}^k |v_j|^{-p}}\left(k - \left(\frac{-\pi\Gamma(-2p)\sin(\pi p)}{\left[\Gamma(-p)\sin\left(\frac{\pi}{2}p\right)\right]^2} - 1\right)\right),$$

$$Var\left(\hat{F}_{p,hm}\right) = F_p^2\frac{1}{k}\left(\frac{-\pi\Gamma(-2p)\sin(\pi p)}{\left[\Gamma(-p)\sin\left(\frac{\pi}{2}p\right)\right]^2} - 1\right) + O\left(\frac{1}{k^2}\right).$$

$$\lim_{p \to 0+} -\frac{2}{\pi}\Gamma(-p)\sin\left(\frac{\pi}{2}p\right) \to 1, \qquad \lim_{p \to 0+} \frac{-\pi\Gamma(-2p)\sin(\pi p)}{\left[\Gamma(-p)\sin\left(\frac{\pi}{2}p\right)\right]^2} - 1 \to 1.$$

Denote this estimator by $\hat{f}_{srp}$ (using $p$ as small as possible), whose variance is $Var\left(\hat{f}_{srp}\right) \approx \frac{f^2}{k}$, which is roughly equivalent to the variance of $\hat{f}$, the unbiased estimator for CRS.

We empirically compared CRS with SRP. Four word vectors were selected; entries of each vector record the numbers of occurrences of the word in $D = 2^{16}$ Web pages. The data are very heavy-tailed. The percentage of zero elements (i.e., sparsity) varies from 58% to 95%.

Figure 2 presents the comparisons. (1): It is possible that CRS may outperform SRP non-negligibly. (2): The variance (3) based on the approximate "conditioning" argument is very accurate. (3): The unbiased estimator $\hat{f}$ is more accurate than $\hat{f}_{mle}$; the latter actually uses one more sample.

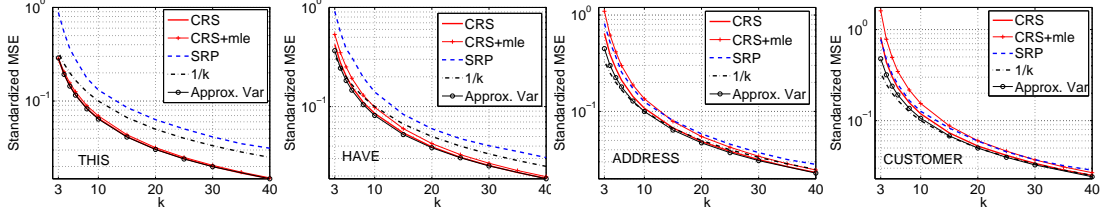

Figure 2: Comparing CRS with SRP for approximating Hamming norms in Web crawl data (four word vectors), using the normalized mean square errors (MSE, normalized by $f^2$). "CRS" and "CRS+mle" respectively correspond to $\hat{f}$ and $\hat{f}_{mle}$, derived in Lemma 1. "SRP" corresponds to the *harmonic mean* estimator of SRP using $p = 0.04$. "$1/k$" is the theoretical asymptotic variance of both CRS and SRP. The curve labeled "Approx. Var" is the approximate variance in (3).

# 5 The Modified CRS Estimation Procedure

The modified CRS estimation procedure is based on the theoretical analysis for using CRS to approximate Hamming norms. Suppose we are interested in the distance between rows $u_1$ and $u_2$ and we have access to sketches $K_1$ and $K_2$. Our suggested "equivalent" sample size $D_s$ would be

$$D_s = \min\{Z_1 - 1, Z_2 - 1\}, \qquad Z_1 = \max(\mathrm{ID}(K_1)), \quad Z_2 = \max(\mathrm{ID}(K_2)). \tag{4}$$

We should not include elements in $K_1$ and $K_2$ whose IDs are larger than $D_s$

Consider $K_1$ and $K_2$ in Figure 1, the modified CRS adopts $D_s = \min(10 - 1, 8 - 1) = \min(9, 7) = 7$. Removing $10\{8\}$ from $K_1$ and $8\{7\}$ from $K_2$, we obtain a sample for $u_1$ and $u_2$:

$$\tilde{u}_{1,1} = 5, \ \tilde{u}_{1,4} = 1, \ \tilde{u}_{1,6} = 7, \qquad \tilde{u}_{2,2} = 9, \ \tilde{u}_{2,3} = 2, \ \tilde{u}_{2,5} = 6.$$

All other sample entries are zero: $\tilde{u}_{1,2} = \tilde{u}_{1,3} = \tilde{u}_{1,5} = \tilde{u}_{1,7} = 0$, $\tilde{u}_{2,1} = \tilde{u}_{2,4} = \tilde{u}_{2,6} = \tilde{u}_{2,7} = 0$.

## 5.1 A Generic Estimator and Approximate Variance

Rigorous theoretical analysis on one pair of sketches is difficult. We resort to the approximate "conditioning" argument using the modified $D_s$ in (4). We consider a generic distance $d_g(u_1, u_2) = \sum_{i=1}^{D} g(u_{1,i}, u_{2,i})$, and assume that, conditioning on $D_s$, the sample $\{\tilde{u}_{1,j}, \tilde{u}_{2,j}\}_{j=1}^{D_s}$ is exactly equivalent to the sample from randomly selected $D_s$ columns without replacement. Under this assumption, an "unbiased" estimator of $d_g(u_1, u_2)$ (and two special cases) would be

$$\hat{d}_g(u_1, u_2) = \frac{D}{D_s} \sum_{i=1}^{D} g(\tilde{u}_{1,j}, \tilde{u}_{2,j}), \quad \hat{d}_p = \frac{D}{D_s} \sum_{j=1}^{D_s} |\tilde{u}_{1,j} - \tilde{u}_{2,j}|^p, \quad \hat{d}_{\chi^2} = \frac{D}{D_s} \sum_{j=1}^{D_s} \frac{(\tilde{u}_{1,j} - \tilde{u}_{2,j})^2}{\tilde{u}_{1,j} + \tilde{u}_{2,j}}.$$

A generic (approximate) variance formula can be obtained as follows:

$$\mathrm{Var}\left(\hat{d}_g(u_1, u_2)|D_s\right) \approx \frac{D - D_s}{D - 1} \times \frac{D^2}{D_s^2} D_s \left(\mathrm{E}\left(g^2(\tilde{u}_{1,j}, \tilde{u}_{2,j})\right) - \mathrm{E}^2\left(g(\tilde{u}_{1,j}, \tilde{u}_{2,j})\right)\right)$$

$$= \frac{D - D_s}{D - 1} \frac{D^2}{D_s^2} D_s \left(\frac{1}{D} \sum_{i=1}^{D} g^2(u_{1,i}, u_{2,i}) - \left(\frac{1}{D} \sum_{i=1}^{D} g(u_{1,i}, u_{2,i})\right)^2\right) = \frac{D}{D - 1} \left(\frac{D}{D_s} - 1\right) \left(d_{g^2} - \frac{d_g^2}{D}\right).$$

$$\mathrm{Var}\left(\hat{d}_g(u_1, u_2)\right) \approx \mathrm{E}\left(\mathrm{Var}\left(\hat{d}_g(u_1, u_2)|D_s\right)\right) = \frac{D}{D - 1} \left(\mathrm{E}\left(\frac{D}{D_s}\right) - 1\right) \left(d_{g^2} - \frac{d_g^2}{D}\right)$$

$$= \frac{D}{D - 1} \left(\mathrm{E}\left(\max\{\frac{D}{Z_1 - 1}, \frac{D}{Z_2 - 1}\}\right) - 1\right) \left(d_{g^2} - \frac{d_g^2}{D}\right)$$

$$\approx \frac{D}{D - 1} \left(\max\left\{\mathrm{E}\left(\frac{D}{Z_1 - 1}\right), \mathrm{E}\left(\frac{D}{Z_2 - 1}\right)\right\} - 1\right) \left(d_{g^2} - \frac{d_g^2}{D}\right)$$

$$= \frac{D}{D - 1} \left(\max\left\{\frac{f_1}{k_1 - 1}, \frac{f_2}{k_2 - 1}\right\} - 1\right) \left(d_{g^2} - \frac{d_g^2}{D}\right). \tag{5}$$

Here, $k_1$ and $k_2$ are the sketch sizes of $K_1$ and $K_2$, respectively, $f_1$ and $f_2$ are the numbers of non-zeros in the original data, $u_1$, $u_2$, respectively. We have used the results in Lemma 1 and a common statistical approximation: $\mathrm{E}(\max(x, y)) \approx \max(\mathrm{E}(x), \mathrm{E}(y))$.

From (5), we know the variance is affected by two factors. If the data are very sparse, i.e., $\max\left\{\frac{f_1}{k_1-1}, \frac{f_2}{k_2-1}\right\}$ is small, then the variance also tends to be small. If the data are heavy-tailed, i.e., $Dd_{g^2} \gg d_g^2$, then the variance tends to be large. Text data are often highly sparse and heavy-tailed; but machine learning applications often need to use the weighted data (i.e., taking logarithm or binary quantization). This is why we expect CRS will be successful in real applications, although it in general does not have the worst-case performance guarantees.

The next two subsections apply CRS to estimating the Hamming distance and the $\chi^2$ distance. Empirical studies [3, 7, 9] have demonstrated that, in text and image data, using the Hamming distance or the $\chi^2$ distance for kernel SVMs achieved good performance.

## 5.2  Estimating the Hamming Distance

Following the definition of Hamming distance in [4]: $h(u_1, u_2) = \sum_{i=1}^{D} 1\{u_{1,i} - u_{2,i} \neq 0\}$, we estimate $h$ using the modified CRS procedure, denoted by $\hat{h}$. The approximate variance (5) becomes

$$\text{Var}\left(\hat{h}\right) \approx \frac{D}{D-1}\left(\max\left\{\frac{f_1}{k_1-1}, \frac{f_2}{k_2-1}\right\} - 1\right)\left(h - \frac{h^2}{D}\right). \tag{6}$$

We also apply SRP using small $p$ and its most accurate harmonic mean estimator[10]. The empirical comparisons in Figure 3 verify two points. (1): CRS can be considerably more accurate than SRP for estimating Hamming distances in [4]. (2): The approximate variance formula (6) is very accurate.

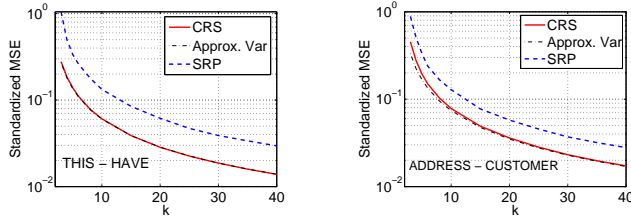

Figure 3: Approximating Hamming distances ($h$) using two pairs of words. The results are presented in terms of the normalized (by $h^2$) MSE. The curves labeled "Approx. Var" correspond to the approximate variance of CRS in (6).

In this example, the seemingly impressive improvement of CRS over SRP is actually due to that we used the definition of Hamming distance in [4]. An alternative definition of Hamming distance is $h(u_1, u_2) = \sum_{i=1}^{D}[1\{u_{1,i} \neq 0 \text{ and } u_{2,i} = 0\} + 1\{u_{1,i} = 0 \text{ and } u_{2,i} \neq 0\}]$, which is basically the $l_p$ distance after a binary term-weighting. As we have commented, if using SRP in dynamic data, term-weighting is not possible; thus we only experimented with the definition in [4].

## 5.3  Estimating the $\chi^2$ Distance

We apply CRS to estimating the $\chi^2$ distance between $u_1$ and $u_2$: $d_{\chi^2}(u_1, u_2) = \sum_{i=1}^{D} \frac{(u_{1,i} - u_{2,i})^2}{u_{1,i} + u_{2,i}}$. According to (5), the estimation variance should be approximately

$$\frac{D}{D-1}\left(\max\left\{\frac{f_1}{k_1-1}, \frac{f_2}{k_2-1}\right\} - 1\right)\left(\sum_{i=1}^{D} \frac{(u_{1,i} - u_{2,i})^4}{(u_{1,i} + u_{2,i})^2} - \frac{d_{\chi^2}^2}{D}\right), \tag{7}$$

which is affected only by the second moments, because $\sum_{i=1}^{D} \frac{(u_{1,i} - u_{2,i})^4}{(u_{1,i} + u_{2,i})^2} \leq \sum_{i=1}^{D}(u_{1,i} + u_{2,i})^2$. There are proved negative results [6] that in the worst-case no efficient algorithms exist for approximating the $\chi^2$ distances. CRS does not provide any worst-case guarantees; its performance relies on the assumption that the data are often reasonably sparse and the second moments should be reasonably bounded in machine learning applications.

Figure 4 presents some empirical study, using the same four words, plus the UCI Dexter data. Even though the four words are fairly common (i.e., not very sparse) and they are heavy-tailed (no term-weighting was applied), CRS still achieved good performance in terms of the normalized MSE (e.g., $\leq 0.1$) at reasonably small $k$. And again, the approximate variance formula (7) is accurate.

Results in the Dexter data set (which is more realistic for machine learning) are encouraging. Only about $k = 10$ is needed to achieve small MSE.

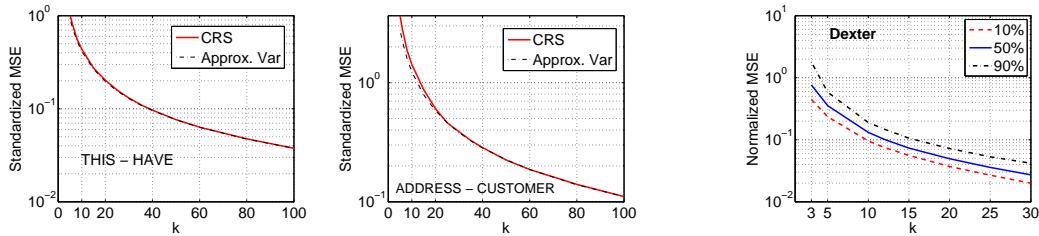

Figure 4: Left two panels: CRS for approximating the $\chi^2$ distance using two pairs of words ($D = 2^{16}$). The curves report the normalized MSE and the approximate variance in (7).
Right-most panel: The Dexter data, $D = 20000$, with 300 data points. We estimate all pairwise (i.e., 44850 pairs) $\chi^2$ distances using CRS. The three curves report the quantiles of normalized MSEs.

## 6 Conclusion

The ubiquitous phenomenon of massive, high-dimensional, and possibly dynamic data, has brought in serious challenges. It is highly desirable to achieve compact data presentation and efficiently computing and retrieving summary statistics, in particular, various types of distances. *Conditional Random Sampling (CRS)* provides a simple and effective mechanism to achieve this goal.

Compared with other "main stream" *sketching* algorithms such as *stable random projections (SRP)*, the major advantage of CRS is that it is "one-sketch-for-all," meaning that the same set of sketches can approximate any linear summary statistics. This would be very convenient in practice.

The major disadvantage of CRS is that it relies heavily on the data sparsity and also on the assumption that in machine learning applications the "worst-case" data distributions are often avoided (e.g., through term-weighting). Also, the theoretical analysis is difficult, despite it is a simple algorithm.

Originally based on a heuristic argument, the preliminary version of CRS, was proposed as a tool for computing pairwise $l_2$ and $l_1$ distances in static data. This paper provides a partial theoretical justification of CRS and various modifications, to make the algorithm more rigorous and to extend CRS for handling dynamic/streaming data. We demonstrate, empirically and theoretically, the effectiveness of CRS in approximating the Hamming norms/distances and the $\chi^2$ distances.

## Acknowledgement

Ping Li is partially supported by grant DMS-0808864 from the National Science Foundation, and a gift from Microsoft. Trevor Hastie was partially supported by grant DMS-0505676 from the National Science Foundation, and grant 2R01 CA 72028-07 from the National Institutes of Health.

## Footnotes

[1] We leave it for particular applications to decide whether an entry updated to zero should be discarded or should be kept in the sketch. In reality, this case does not occur often. For example, the most important type of data streams[15] is "insertion-only," meaning that the values will never decrease.

## References

[1] Charu C. Aggarwal, Jiawei Han, Jianyong Wang, and Philip S. Yu. On demand classification of data streams. In *KDD*, 503–508, 2004.

[2] Léon Bottou, Olivier Chapelle, Dennis DeCoste, and Jason Weston, editors. *Large-Scale Kernel Machines*. The MIT Press, 2007.

[3] Olivier Chapelle, Patrick Haffner, and Vladimir N. Vapnik. Support vector machines for histogram-based image classification. *IEEE Trans. Neural Networks*, 10(5):1055–1064, 1999.

[4] Graham Cormode, Mayur Datar, Piotr Indyk, and S. Muthukrishnan. Comparing data streams using hamming norms (how to zero in). *IEEE Transactions on Knowledge and Data Engineering*, 15(3):529–540, 2003.

[5] Carlotta Domeniconi and Dimitrios Gunopulos. Incremental support vector machine construction. In *ICDM*, pages 589–592, 2001.

[6] Sudipto Guha, Piotr Indyk, and Andrew McGregor. Sketching infomration divergence. In *COLT*, pages 424–438, 2007.

[7] M. Hein and O. Bousquet. Hilbertian metrics and positive definite kernels on probability measures. In *AISTATS*, pages 136–143, 2005.

[8] Piotr Indyk. Stable distributions, pseudorandom generators, embeddings, and data stream computation. *J. of ACM*, 53(3):307–323, 2006.

[9] Yugang Jiang, Chongwah Ngo, and Jun Yang. Towards optimal bag-of-features for object categorization and semantic video retrieval. In *CIVR*, pages 494–501, 2007.

[10] Ping Li. Estimators and tail bounds for dimension reduction in $l_\alpha$ ($0 < \alpha \le 2$) using stable random projections. In *SODA*, 2008.

[11] Ping Li. Compressed Counting. In *SODA*, 2009.

[12] Ping Li and Kenneth W. Church. A sketch algorithm for estimating two-way and multi-way Associations. *Computational Linguistics*, 33(3):305-354, 2007. Preliminary results appeared in HLT/EMNLP, 2005.

[13] Ping Li, Kenneth W. Church, and Trevor J. Hastie. Conditional random sampling: A sketch-based sampling technique for sparse data. In *NIPS*, pages 873–880, 2007.

[14] Ping Li. Computationally efficient estimators for dimension reductions using stable random projections. In *ICDM*, 2008.

[15] S. Muthukrishnan. Data streams: Algorithms and applications. *Found. and Trends in Theoretical Computer Science*, 1:117–236, 2 2005.

[16] John C. Platt. Using analytic QP and sparseness to speed training of support vector machines. In *NIPS*, pages 557–563, 1998.

[17] Bernhard Schölkopf and Alexander J. Smola. *Learning with Kernels*. The MIT Press, 2002.
